# Topographic Map Formation by Silicon Growth Cones

**Brian Taba and Kwabena Boahen**
Department of Bioengineering
University of Pennsylvania
Philadelphia, PA 19104
*{btaba,kwabena}@neuroengineering.upenn.edu*

## Abstract

We describe a self-configuring neuromorphic chip that uses a model of activity-dependent axon remodeling to automatically wire topographic maps based solely on input correlations. Axons are guided by growth cones, which are modeled in analog VLSI for the first time. Growth cones migrate up neurotropin gradients, which are represented by charge diffusing in transistor channels. Virtual axons move by rerouting address-events. We refined an initially gross topographic projection by simulating retinal wave input.

## 1   Neuromorphic Systems

Neuromorphic engineers are attempting to match the computational efficiency of biological systems by morphing neurocircuitry into silicon circuits [1]. One of the most detailed implementations to date is the silicon retina described in [2]. This chip comprises thirteen different cell types, each of which must be individually and painstakingly wired. While this circuit-level approach has been very successful in sensory systems, it is less helpful when modeling largely unelucidated and exceedingly plastic higher processing centers in cortex.

Instead of an explicit blueprint for every cortical area, what is needed is a developmental rule that can wire complex circuits from minimal specifications. One candidate is the famous "cells that fire together wire together" rule, which strengthens excitatory connections between coactive presynaptic and postsynaptic cells. We implemented a self-rewiring scheme of this type in silicon, taking our cue from axon remodeling during development.

## 2   Growth Cones

During development, the brain wires axons into a myriad of topographic projections between regions. Axonal projections initially organize independent of neural activity, establishing a coarse spatial order based on gradients of substrate-bound molecules laid down by local gene expression. These gross topographic projections are refined and maintained by subsequent neuronal spike activity, and can reroute

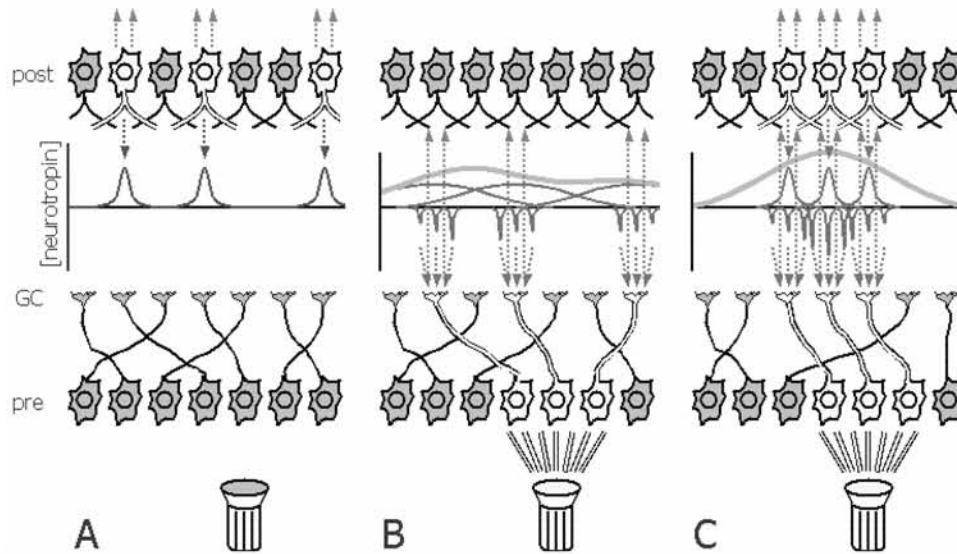

**Figure 1: A.** Postsynaptic activity is transmitted to the next layer (up arrows) and releases neurotropin into the extracellular medium (down arrows). **B.** Presynaptic activity excites postsynaptic dendrites (up arrows) and triggers neurotropin uptake by active growth cones (down arrows). Each growth cone samples the neurotropin concentration at several spatial locations, measuring the gradient across the axon terminal. Growth cones move toward higher neurotropin concentrations. **C.** Axons that fire at the same time migrate to the same place.

themselves if their signal source changes. In such cases, axons abandon obsolete territory and invade more promising targets [3].

An axon grows by adding membrane and microtubule segments to its distal tip, an amoeboid body called a growth cone. Growth cones extend and retract fingers of cytoplasm called filopodia, which are sensitive to local levels of guidance chemicals in the surrounding medium. Candidate guidance chemicals include BDNF and NO, whose release can be triggered by action potentials in the target neuron [4].

Our learning rule is based on an activity-derived diffusive chemical that guides growth cone migration. In our model, this neurotropin is released by spiking neurons and diffuses in the extracellular medium until scavenged by glia or bound by growth cones (Figure 1A). An active growth cone compares amounts of neurotropin bound to each of its filopodia in order to measure the local gradient (Figure 1B). The growth cone then moves up the gradient, dragging the axon behind it. Since neurotropin is released by postsynaptic activity and axon migration is driven by presynaptic activity, this rule translates temporal coincidence into spatial coincidence (Figure 1C).

For topographic map formation, this migration rule requires temporal correlations in the presynaptic plane to reflect neighborhood relations. We supply such correlations by simulating retinal waves, spontaneous bursts of action potentials that sweep across the ganglion cell layer in the developing mammalian retina. Retinal waves start at random locations and spread over a limited domain before fading away, eventually tiling the entire retinal plane [5]. Axons participating in the same retinal

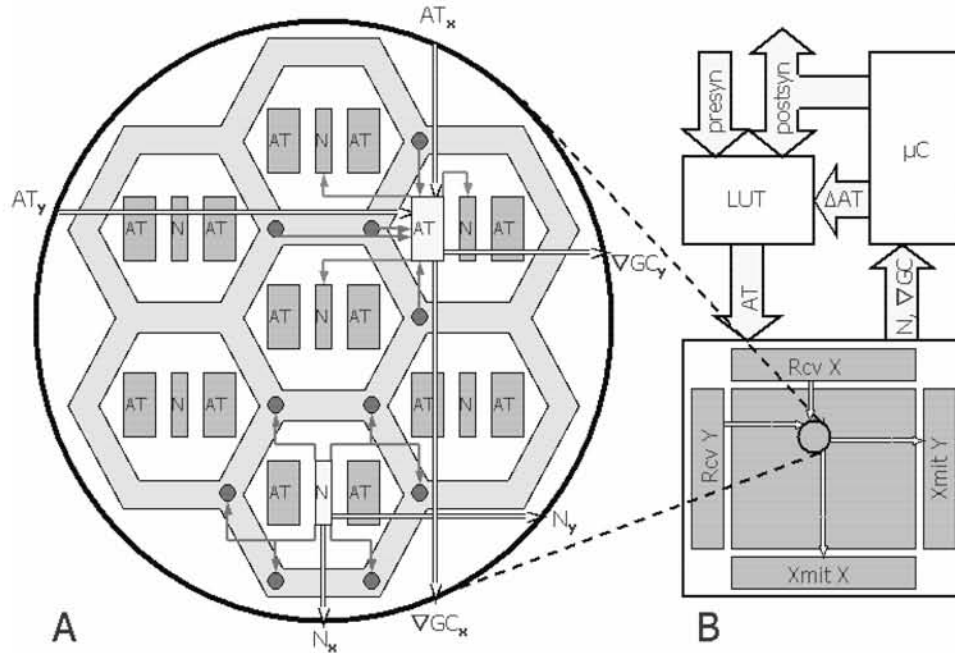

**Figure 2: A.** Chip block diagram. Axon terminal (AT) and neuron (N) circuits are arrayed hexagonally, surrounded by a continuous charge-diffusing lattice. An active axon terminal ($AT_{x,y}$) excites the three adjacent neurons and its growth cone samples neurotropin from four adjacent lattice nodes. The growth cone sends the measured gradient direction off-chip ($\nabla GC_{x,y}$). An active postsynaptic neuron ($N_{x,y}$) releases neurotropin into the six surrounding lattice nodes and sends its spike off-chip. **B.** System block diagram. Presynaptic neurons send spikes to the lookup table (LUT), which routes them to axon terminal coordinates (AT) on-chip. Chip output filters through a microcontroller ($\mu C$) that translates gradient measurements ($\nabla GC$) into LUT updates ($\Delta AT$). Postsynaptic activity (N) may be returned to the LUT as recurrent excitation and also passed on to the next stage of the system.

wave migrate to the same postsynaptic neighborhood, since neurotropin concentration is maximized when every cell that fires at the same time releases neurotropin at the same place.

To prevent all of the axons from collapsing onto a single postsynaptic target, we enforce a strictly constant synaptic density. We have a fixed number of synaptic sites, each of which can be occupied by one and only one presynaptic afferent. An axon terminal moves from one synaptic site to another by swapping places with the axon already occupying the desired location. Learning occurs only in the point-to-point wiring diagram; synaptic weights are identical and unchanging.

## 3   System Architecture

We have fabricated and tested a first-generation neurotropin chip, Neurotrope1, that implements retrograde transmission of a diffusive factor from postsynaptic neurons to presynaptic afferents (Figure 2A). The 11.5 mm$^2$ chip was fabricated through MOSIS using the TSMC 0.35μm process, and includes a 40 x 20 array of growth cones interleaved with a 20 x 20 array of neurons. The chip receives and transmits

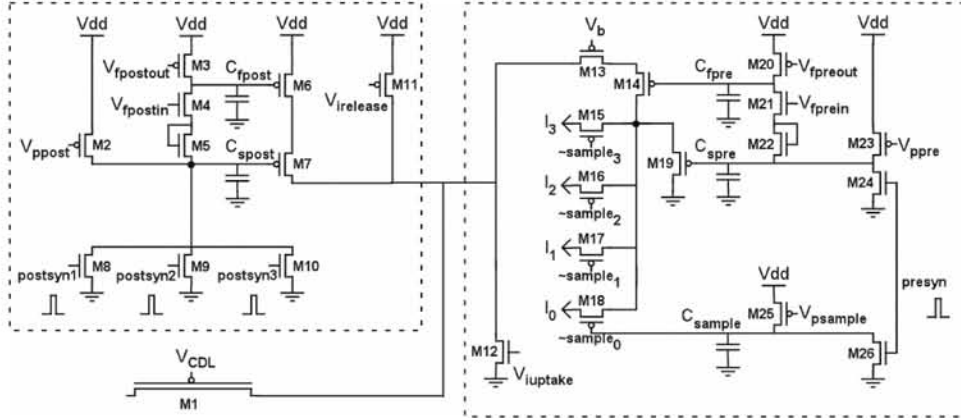

**Figure 3:** Neurotropin circuit diagram. Postsynaptic activity gates neurotropin release (left box) and presynaptic activity gates neurotropin uptake (right box).

spike coordinates encoded as address-events, permitting ready interface with other spike-based chips that obey this standard [6]. Virtual wiring [7] is realized with a look-up table (LUT) stored in a separate content-addressable memory (CAM) that is controlled by an Ubicom SX52 microcontroller (Figure 2B).

The core of the chip consists of an array of axon terminals that target a second array of neurons, all surrounded by a monolithic pFET channel laid out as a hexagonal lattice, representing a two-dimensional extracellular medium. An activated axon terminal generates postsynaptic potentials in all the fixed-radius dendritic arbors that span its location, as modeled by a diffusor network [8]. Once the membrane potential crosses a threshold, the neuron fires, transmitting its coordinates off-chip and simultaneously releasing neurotropin, represented as charge spreading within the lattice. Neurotropin diffuses spatially until removed by either an activity-independent leak current or an active axon terminal.

An axon terminal senses the local extracellular neurotropin gradient by draining charge from its own node on the hexagonal lattice and from the three immediately adjacent nodes. Charge from the four locations is integrated on independent capacitors, which race to cross threshold first. The winner of this latency competition transmits a set of coordinates that uniquely identify the location and direction of the measured gradient. We use the neuron circuit described in [9] to integrate neurotropin as well as dendritic potentials.

Coordinates transmitted off-chip thus fall into two categories: neuron spikes that are routed through the LUT, and gradient directions that are used to update entries in the LUT. An axon migrates simply by looking up the entry in the table corresponding to the site it wants to occupy and swapping that address with that of its current location. Subsequent spikes are routed to the new coordinates. Thus, although the physical axon terminal circuits are immobilized in silicon, the virtual axons are free to move within the postsynaptic plane.

## 3.1 Neurotropin circuit

Neurotropin in the extracellular medium is represented by charge in the hexagonal charge-diffusing lattice M1 (Figure 3). $V_{CDL}$ sets the maximum amount of charge M1 can hold. The total charge in M1 is determined by circuits that implement

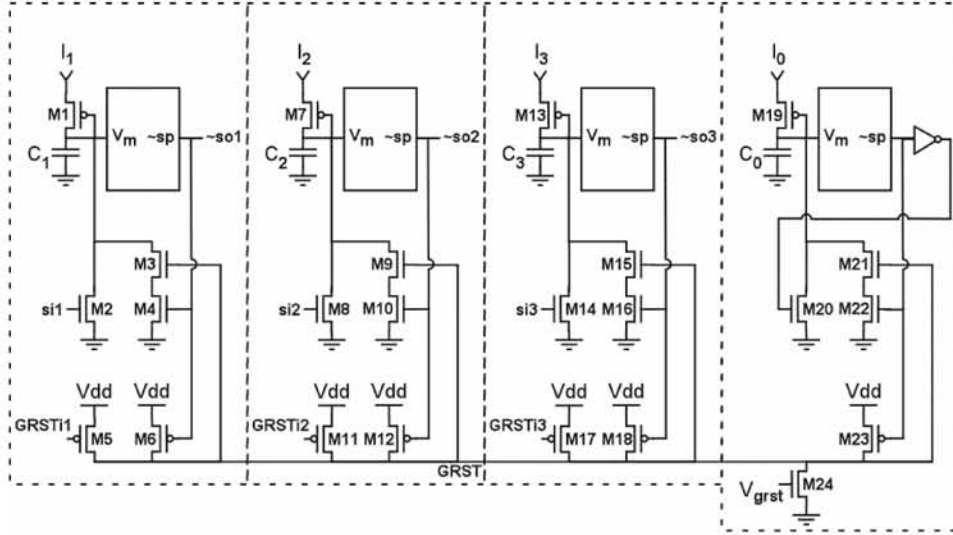

**Figure 4:** Latency competition circuit diagram. A growth cone integrates neurotropin samples from its own location (right box) and the three neighboring locations (left three boxes). The first location to accumulate a threshold of charge resets its three competitors and signals its identity off-chip.

activity-dependent neurotropin release and uptake. In addition, M11 and M12 provide a path for activity-independent release and uptake.

Postsynaptic activity triggers neurotropin release, as implemented by the circuit in the left box of Figure 3. Spikes from any of the three neighboring postsynaptic neurons pull $C_{spost}$ to ground, opening M7 and discharging $C_{fpost}$ through M4 and M5. As $C_{fpost}$ falls, M6 opens, establishing a transient path from $V_{dd}$ to M1 that injects charge into the hexagonal lattice. Upon termination of the postsynaptic spike, $C_{spost}$ and $C_{fpost}$ are recharged by decay currents through M2 and M3. $V_{ppost}$ and $V_{fpostout}$ are chosen such that $C_{spost}$ relaxes faster than $C_{fpost}$, permitting $C_{fpost}$ to integrate several postsynaptic spikes and facilitate charge injection if spikes arrive in a burst rather than singly. $V_{fpostin}$ determines the contribution of an individual spike to the facilitation capacitor $C_{fpost}$.

Presynaptic activity triggers neurotropin uptake, as implemented by the circuit in the right box of Figure 3. Charge is removed from the hexagonal lattice by a facilitation circuit similar to that used for postsynaptic release. A presynaptic spike targeted to the axon terminal pulls $C_{spre}$ to ground through M24. $C_{spre}$, in turn, drains charge from $C_{fpre}$ through M21 and M22. $C_{fpre}$ removes charge from the hexagonal lattice through M14, up to a limit set by M13, which prevents the hexagonal lattice from being completely drained in order to avoid charge trapping. Current from M14 is divided between five possible sinks. Depending on presynaptic activation, up to four axon terminals may sample a fraction of this current through M15-18; the remainder is shunted to ground through M19 in order to prevent a single presynaptic event from exerting undue influence on gradient measurements. The current sampled by the axon terminal at its own site is gated by $\sim sample_0$, which is pulled low by a presynaptic spike through M26 and subsequently recovers through M25. Identical circuits in the other axon terminals generate signals $\sim sample_1$, $\sim sample_2$, and $\sim sample_3$. Sample currents $I_0$, $I_1$, $I_2$, and $I_3$ are routed to latency competition circuits in the four adjacent axon terminals.

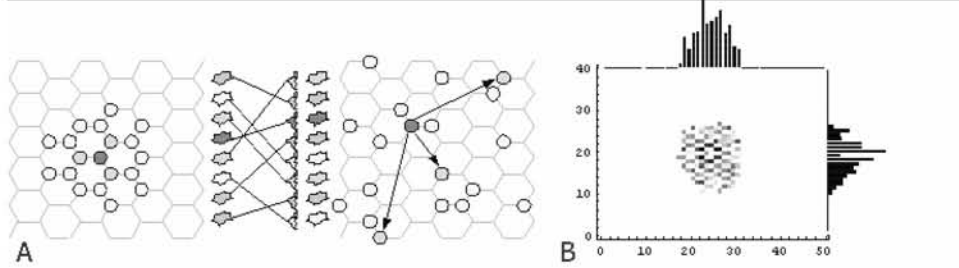

**Figure 5:** Retinal stimulus and cortical attractor. **A.** Randomly centered patches of active retinal cells (left) excite cortical targets (right). **B.** Density plot of a single mobile growth cone initialized in a static topographic projection. Histograms bin column ($\sigma$=3.27) and row ($\sigma$=3.79) coordinates observed (n=800).

## 3.2 Latency competition circuit

Each axon terminal measures the local neurotropin gradient by sampling a fraction of the neurotropin present at its own site, location 0, and the three immediately adjacent nodes on the hexagonal lattice, locations 1-3. Charge drained from the hexagonal lattice at these four sites is integrated on a separate capacitor for each location. The first capacitor to reach the threshold voltage wins the race, resetting itself and all of its competitors and signaling its victory off-chip.

In the circuit that samples neurotropin from location 1 (left box of Figure 4), charge pulses $I_1$ arrive through diode M1 and accumulate on capacitor $C_1$ in an integrate-and-fire circuit described in [9]. Upon crossing threshold this circuit transmits a swap request $\sim so1$, resets its three competitors by using M6 to pull the shared reset line *GRST* high, and disables M4 to prevent *GRST* from using M3 to reset $C_1$. The swap request $\sim so1$ remains low until acknowledged by *si1*, which discharges $C_1$ through M2. During the time that $\sim so1$ is low, the other three capacitors are shunted to ground by *GRST*, preventing late arrivals from corrupting the declared gradient measurement before it has been transmitted off-chip. $C_1$ being reset releases *GRST* to relax to ground through M24 with a decay time determined by $V_{grst}$.

$C_1$ is also reset if the neighboring axon terminal initiates a swap. *GRSTi1* is pulled low if either the axon terminal at location 1 decides to move to location 0 or the axon terminal at location 0 decides to move to location 1. The accumulated neurotropin samples at both locations become obsolete after the exchange, and are therefore discarded when *GRST* is pulled high through M5. Identical circuits sample neurotropin from locations 2 and 3 (center two boxes of Figure 4).

If $C_0$ (right box of Figure 4) wins the latency competition, the axon terminal decides that its current location is optimal and therefore no action is required. In this case, no off-chip communication occurs and $C_0$ immediately resets itself and its three rivals. Thus, the location 0 circuit is identical to those of locations 1-3 except that the inverted spike is fed directly back to the reset transistor M20 instead of to a communication circuit. Also, there is no *GRSTi0* transistor since there is no swap partner.

## 4 Results

We drove the chip with a sequence of randomly centered patches of presynaptic activity meant to simulate retinal waves. Each patch consisted of 19 adjacent presynaptic cells: a randomly selected presynaptic cell and its nearest, next-nearest,

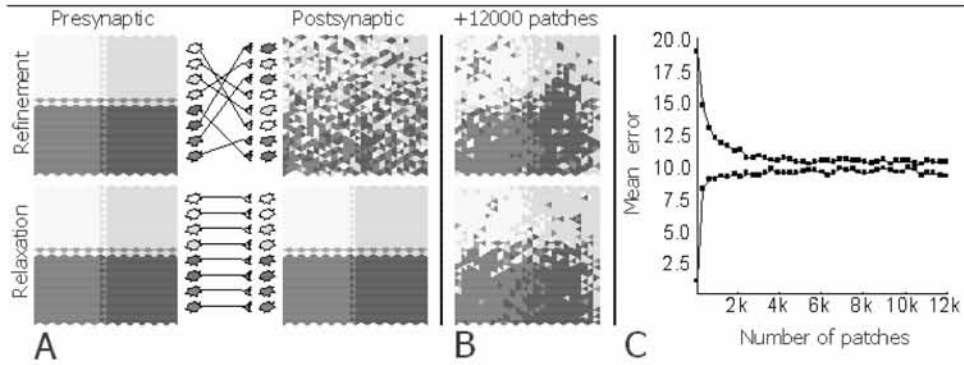

**Figure 6:** Topographic map evolution. **A.** Initial maps. Axon terminals in the postsynaptic plane (right) are dyed according to the presynaptic coordinates of their cell body (left). Top row: Coarse initial map. Bottom row: Perfect initial map. **B.** Postsynaptic plane after 12000 patch presentations. **C.** Map error in units of average postsynaptic distance between axon terminals of presynaptic neighbors. Top line: refinement of coarse initial map; bottom line: relaxation of perfect initial map.

and third-nearest presynaptic neighbors on a hexagonal grid (Figure 5A). Every patch participant generated a burst of 8192 spikes, which were routed to the appropriate axon terminal circuit according to the connectivity map stored in the CAM. About 100 patches were presented per minute.

To establish an upper performance bound, we initialized the system with a perfectly topographic projection and generated bursts from the same retinal patch, holding all growth cones static except for the one projected from the center of the patch, which was free to move over the entire cortical plane. Over 800 min, the single mobile growth cone wandered within the cortical area of the patch (Figure 5B), suggesting that the patch radius limits maximum sustainable topography even in the ideal case.

To test this limit empirically, we generated an initial connectivity map by starting with a perfectly topographic projection and executing a sequence of $(N/2)^2$ swaps between a randomly chosen axon terminal and one of its randomly chosen postsynaptic neighbors, where $N$ is the number of axon terminals used. We opted for a fanout of 1 and full synaptic site occupancy, so 480 presynaptic cells projected axons to 480 synaptic sites. (One side of the neuron array exhibited enhanced excitability, apparently due to noise on the power rails, so the 320 synaptic sites on that side were abandoned.) The perturbed connectivity map preserved a loose global bias, representing the formation of a coarse topographic projection from activity-independent cues. This new initial map was then allowed to evolve according to the swap requests generated by the chip. After approximately 12000 patches, a refined topographic projection reemerged (Figure 6A,B).

To investigate the dynamics of topographic refinement, we defined the error for a single presynaptic cell to be the average of the postsynaptic distances between the axon terminals projected by the cell body and its three immediate presynaptic neighbors. A cell in a perfectly topographic projection would therefore have unit error. The error drops quickly at the beginning of the evolution as local clumps of correlated axon terminals crystallize. Further refinement requires the disassembly of locally topographic crystals that happened to nucleate in a globally inconvenient location. During this later phase, the error decreases slowly toward an asymptote. To evaluate this limit we seeded the system with a perfect projection and let it relax

to a sustainable degree of topography, which we found to have an error of about 10 units (Figure 6C).

## 5  Discussion

Our results demonstrate the feasibility of a spike-based neuromorphic learning system based on principles of developmental plasticity. This neurotropin chip lends itself readily to more ambitious multichip systems incorporating silicon retinae that could be used to automatically wire ocular dominance columns and orientation-selectivity maps when driven by spatiotemporal correlations among neurons of different origin (e.g. left eye/right eye) or type (ON/OFF).

A related model of chemical-driven developmental plasticity posits an activity-dependent competition for a local sustenance factor, or neurotrophin. Axon weights saturate at neurotrophin-rich locations and vanish at neurotrophin-starved locations, pruning a dense initial arbor until only the final circuit remains [10]. By contrast, in our chemotaxis model, a handful of growth cone-guided wires rearrange themselves by moving through locations at which they had no initial presence. These two mechanisms could plausibly complement each other: noisy gradient measurements establish an initial axonal arbor that can then be pruned to eliminate outliers and refine local topography. We can use a similar approach to improve our silicon maps.

## Acknowledgments

We would like to thank K. Hynna and K. Zaghloul for assistance with fabrication and testing. This project was funded in part by the David and Lucille Packard Foundation and the NSF/BITS program (EIA0130822). B.T. received support from the Dolores Zohrab Liebmann Foundation.

## References

[1]  C. Mead (1990) Neuromorphic electronic systems. *IEEE Proc*, 78(10): 1629-1636.

[2]  K.A. Zaghloul (2002) A silicon implementation of a novel model for retinal processing. PhD thesis, University of Pennsylvania.

[3]  M. Sur and C.A. Leamy (2001) Development and plasticity of cortical areas and networks. *Nat Rev Neurosci*, 2:251-262.

[4]  E.J. Huang and L.F. Reichardt (2001) Neurotrophins: roles in neuronal development and function. *Annu Rev Neurosci*, 24:677-736.

[5]  M.B. Feller, D.A. Butts, H.L. Aaron, D.S. Rokhsar, and C.J. Shatz (1997) Dynamic processes shape spatiotemporal properties of retinal waves. *Neuron*, 19:293-306.

[6]  K.A. Boahen (2000) Point-to-point connectivity between neuromorphic chips using address-events. *IEEE Transactions on Circuits and Systems II*, 47:416-434.

[7]  J.G. Elias (1993) Artificial dendritic trees. *Neural Comp*, 5:648-663.

[8]  K.A. Boahen and A.G. Andreou (1991) A contrast-sensitive silicon retina with reciprocal synapses. *Advances in Neural Information Processing Systems 4*, J.E. Moody and R.P. Lippmann, eds., pp 764-772, Morgan Kaufman, San Mateo, CA.

[9]  E. Culurciello, R. Etienne-Cummings, and K. Boahen (2001) Arbitrated address event representation digital image sensor. *IEEE International Solid State Circuits Conference*, pp 92-93.

[10] T. Elliott and N.R. Shadbolt (1999) A neurotrophic model of the development of the retinogeniculocortical pathway induced by spontaneous retinal waves. *J Neurosci*, 19:7951-7970.
